# A General Projection Property for Distribution Families

**Yao-Liang Yu    Yuxi Li    Dale Schuurmans    Csaba Szepesvári**
Department of Computing Science
University of Alberta
Edmonton, AB, T6G 2E8 Canada
{yaoliang,yuxi,dale,szepesva}@cs.ualberta.ca

## Abstract

Surjectivity of linear projections between distribution families with fixed mean and covariance (regardless of dimension) is re-derived by a new proof. We further extend this property to distribution families that respect additional constraints, such as symmetry, unimodality and log-concavity. By combining our results with classic univariate inequalities, we provide new worst-case analyses for natural risk criteria arising in classification, optimization, portfolio selection and Markov decision processes.

## 1 Introduction

In real applications, the model of the problem at hand inevitably embodies some form of uncertainty: the parameters of the model are usually (roughly) estimated from data, which themselves can be uncertain due to various kinds of noises. For example, in finance, the return of a financial product can seldom be known exactly beforehand. Despite this uncertainty, one still usually has to take action in the underlying application. However, due to uncertainty, any attempt to behave "optimally" in the world must take into account plausible alternative models.

Focusing on problems where uncertain data/parameters are treated as random variables and the model consists of a joint distribution over these variables, we initially assume prior knowledge that the first and second moments of the underlying distribution are known, but the distribution is otherwise arbitrary. A parametric approach to handling uncertainty in such a setting would be to fit a specific parametric model to the known moments and then apply stochastic programming techniques to solve for an optimal decision. For example, fitting a Gaussian model to the constraints would be a popular choice. However, such a parametric strategy can be too bold, hard to justify, and might incur significant loss if the fitting distribution does not match the true underlying distribution very well. A conservative, but more robust approach would be to take a decision that was "protected" in the worst-case sense; that is, behaves optimally assuming that nature has the freedom to choose an adverse distribution. Such a *minimax* formulation has been studied in several fields [1; 2; 3; 4; 5; 6] and is also the focus of this paper. Although Bayesian optimal decision theory is a rightfully well-established approach for decision making under uncertainty, minimax has proved to be a useful alternative in many domains, such as finance, where it is difficult to formulate appropriate priors over models. In these fields, minimax formulation combined with stochastic programming [7] have been extensively studied and successfully applied.

We make a contribution to minimax probability theory and apply the results to problems arising in four different areas. Specifically, we generalize a classic result on the linear projection property of distribution families: we show that any linear projection between distribution families with fixed mean and covariance, regardless of their dimensions, is *surjective*. That is, given any matrix $X$ and any random vector $\mathbf{r}$ with mean $X^T\mu$ and covariance $X^T\Sigma X$, one can always find another random vector $\mathbf{R}$ with mean $\mu$ and covariance $\Sigma$ such that $X^T\mathbf{R} = \mathbf{r}$ (almost surely). Our proof imposes no conditions on the deterministic matrix $X$, hence extends the classic projection result in [6], which assumes $X$ is a vector. We furthermore extend this surjective property to some restricted distribution

families, which allows additional prior information to be incorporated and hence less conservative solutions to be obtained. In particular, we prove that surjectivity of linear projections remains to hold for distribution families that are additionally symmetric, log-concave, or symmetric linear unimodal. In each case, our proof strategy allows one to construct the worst-case distribution(s).

An immediate application of these results is to reduce the worst-case analysis of multivariate expectations to the univariate (or reduced multivariate) ones, which have been long studied and produced many fruitful results. In this direction, we conduct worst-case analyses of some common restricted distribution families. We illustrate our results on problems that incorporate a classic worst case *value-at-risk* constraint: minimax probability classification [2]; chance constrained linear programming (CCLP) [3]; portfolio selection [4]; and Markov decision processes (MDPs) with reward uncertainty [8]. Although some of the results we obtain have been established in the respective fields [2; 3; 4], we unify them through a much simpler proof strategy. Additionally, we provide extensions to other constrained distribution families, which makes the minimax formulation less conservative in each case. These results are then extended to the more recent *conditional value-at-risk* constraint, and new bounds are proved, including a new bound on the survival function for symmetric unimodal distributions.

## 2 A General Projection Property

First we establish a generalized linear projection property for distribution families. The key application will be to reduce worst-case multivariate stochastic programming problems to lower dimensional equivalents; see Corollary 1. Popescu [6] has proved the special case of reduction to one dimension, however we provide a simpler proof that can be more easily extended to other distribution families[1].

Let $(\mu, \Sigma)$ denote the family of distributions sharing common mean $\mu$ and covariance $\Sigma$, and let $\mu_X = X^T \mu$ and $\Sigma_X = X^T \Sigma X$. Below we denote random variables by boldface letters, and use $\mathbb{I}$ to denote the identity matrix. We use $\dagger$ to denote the pseudo-inverse.

**Theorem 1 (General Projection Property (GPP))** *For all $\mu$, $\Sigma \succeq 0$, and $X \in \mathbb{R}^{m \times d}$, the projection $X^T \mathbf{R} = \mathbf{r}$ from $m$-variate distributions $\mathbf{R} \sim (\mu, \Sigma)$ to $d$-variate distributions $\mathbf{r} \sim (\mu_X, \Sigma_X)$ is* ***surjective*** *and many-to-one. That is, every $\mathbf{r} \sim (\mu_X, \Sigma_X)$ can be obtained from some $\mathbf{R} \sim (\mu, \Sigma)$ via $X^T \mathbf{R} = \mathbf{r}$ (almost surely).*

**Proof**: The proof is constructive. Given a $\mathbf{r} \sim (\mu_X, \Sigma_X)$, we can construct a pre-image $\mathbf{R}$ by letting $\mathbf{R} = \Sigma X \Sigma_X^\dagger \mathbf{r} + (\mathbb{I}_m - \Sigma X \Sigma_X^\dagger X^T) \mathbf{M}$, where $\mathbf{M} \sim (\mu, \Sigma)$ is independent of $\mathbf{r}$, for example, one can choose $\mathbf{M}$ as a Gaussian random vector. It is easy to verify that $\mathbf{R} \sim (\mu, \Sigma)$ and $X^T \mathbf{R} = \Sigma_X \Sigma_X^\dagger \mathbf{r} + (\mathbb{I}_m - \Sigma_X \Sigma_X^\dagger) X^T \mathbf{M} = \mathbf{r}$. The last equality holds since $(\mathbb{I}_m - \Sigma_X \Sigma_X^\dagger) \mathbf{r} = (\mathbb{I}_m - \Sigma_X \Sigma_X^\dagger) X^T \mathbf{M}$ (the two random vectors on both sides have the same mean and zero covariance). Note that since $\mathbf{M}$ can be chosen arbitrarily in $(\mu, \Sigma)$, the projections are always many-to-one. ∎

Although this establishes the general result, we extend it to distribution families under additional constraints below. That is, one often has additional prior information about the underlying distribution, such as symmetry, unimodality, and/or support. In such cases, if a general linear projection property can still be shown to hold, the additional assumptions can be used to make the minimax approach less conservative in a simple, direct manner. We thus consider a number of additionally restricted distribution families.

**Definition 1** *A random vector $\mathbf{X}$ is called (centrally)* ***symmetric*** *about $\mu$, if for all vectors $x$, $\Pr(\mathbf{X} \geq \mu + x) = \Pr(\mathbf{X} \leq \mu - x)$. A univariate random variable is called* ***unimodal*** *about $a$ if its cumulative distribution function (c.d.f.) is convex on $(-\infty, a]$ and concave on $[a, \infty)$. A random vector $\mathbf{X}$ is called* ***log-concave*** *if its c.d.f. is log-concave. A random $m$-vector $\mathbf{X}$ is called* ***linear unimodal*** *about $0_m$ if for all $a \in \mathbb{R}^m$, $a^T \mathbf{X}$ is (univariate) unimodal about 0.*

Let $(\mu, \Sigma)_S$ denote the family of distributions in $(\mu, \Sigma)$ that are additionally symmetric about $\mu$, and similarly, let $(\mu, \Sigma)_L$ denote the family of distributions that are additionally log-concave, and

let $(\mu, \Sigma)_{SU}$ denote the family of distributions that are additionally symmetric *and* linear unimodal about $\mu$. For each of these restricted families, we require the following properties to establish our next main result.

**Lemma 1** *(a) If random vector $\mathbf{X}$ is symmetric about 0, then $A\mathbf{X} + \mu$ is symmetric about $\mu$. (b) If $\mathbf{X}$, $\mathbf{Y}$ are independent and both symmetric about 0, $\mathbf{Z} = \mathbf{X} + \mathbf{Y}$ is also symmetric about 0.*

Although once misbelieved, it is now clear that the convolution of two (univariate) unimodal distributions need *not* be unimodal. However, for symmetric, unimodal distributions we have

**Lemma 2 ([10] Theorem 1.6)** *If two independent random variables $\mathbf{x}$ and $\mathbf{y}$ are both symmetric and unimodal about 0, then $\mathbf{z} = \mathbf{x} + \mathbf{y}$ is also unimodal about 0.*

There are several non-equivalent extensions of unimodality to multivariate random variables. We consider two specific (multivariate) unimodalities in this paper: log-concave and linear unimodal.[2]

**Lemma 3 ([10] Lemma 2.1, Theorem 2.4, Theorem 2.18)**
*1. Linearity: If random $m$-vector $\mathbf{X}$ is log-concave, $a^T\mathbf{X}$ is also log-concave for all $a \in \mathbb{R}^m$.*

*2. Cartesian Product: If $\mathbf{X}$ and $\mathbf{Y}$ are log-concave, then $\mathbf{Z} = \begin{bmatrix} \mathbf{X} \\ \mathbf{Y} \end{bmatrix}$ is also log-concave .*

*3. Convolution: If $\mathbf{X}$ and $\mathbf{Y}$ are independent and log-concave, then $\mathbf{Z} = \mathbf{X} + \mathbf{Y}$ is also log-concave.*

Given the above properties, we can now extend Theorem 1 to $(\mu, \Sigma)_S$, $(\mu, \Sigma)_L$ and $(\mu, \Sigma)_{SU}$.

**Theorem 2 (GPP for Symmetric, Log-concave, and Symmetric Linear Unimodal Distributions)**
*For all $\mu$, $\Sigma \succeq 0$ and $X \in \mathbb{R}^{m \times d}$, the projection $X^T\mathbf{R} = \mathbf{r}$ from $m$-variate $\mathbf{R} \sim (\mu, \Sigma)_S$ to $d$-variate $\mathbf{r} \sim (\mu_X, \Sigma_X)_S$ is **surjective** and many-to-one. The same is true for $(\mu, \Sigma)_L$ and $(\mu, \Sigma)_{SU}$.[3]*

**Proof:** The proofs follow the same basic outline as Theorem 1 except that in the first step we now choose $\mathbf{N} \sim (0_m, \mathbb{I}_m)_S$ or $(0_m, \mathbb{I}_m)_L$ or $(0_m, \mathbb{I}_m)_{SU}$. Then, respectively, symmetry of the constructed $\mathbf{R}$ follows from Lemma 1; log-concavity of $\mathbf{R}$ follows from Lemma 3; and linear unimodality of $\mathbf{R}$ follows from the definition and Lemma 2. The maps remain many-to-one. ∎

An immediate application of the general projection property is to reduce worst-case analyses of multivariate expectations to the univariate case. Note that in the following corollary, the optimal distribution of $\mathbf{R}$ can be easily constructed from the optimal distribution of $\mathbf{r}$.

**Corollary 1** *For any matrix $X$ and any function $g(\cdot)$ (including in particular when $X$ is a vector)*

$$\sup_{\mathbf{R} \sim (\mu, \Sigma)} \mathbb{E}[g(X^T\mathbf{R})] \quad = \quad \sup_{\mathbf{r} \sim (X^T\mu, X^T\Sigma X)} \mathbb{E}[g(\mathbf{r})]. \tag{1}$$

*The equality continues to hold if we restrict $(\mu, \Sigma)$ to $(\mu, \Sigma)_S$, $(\mu, \Sigma)_L$, or $(\mu, \Sigma)_{SU}$ respectively.*

**Proof:** It is obvious that the right hand side is an upper bound on the left hand side, since for every $\mathbf{R} \sim (\mu, \Sigma)$ there exists an $\mathbf{r} \sim (X^T\mu, X^T\Sigma X)$ given by $\mathbf{r} = X^T\mathbf{R}$. Similarly for $(\mu, \Sigma)_S$, $(\mu, \Sigma)_L$, and $(\mu, \Sigma)_{SU}$. However, given Theorems 1 and 2, one can then establish the converse.[4] ∎

## 3 Application to Worst-case Value-at-risk

We now apply these projection properties to analyze the worst case value-at-risk (VaR) —a useful risk criterion in many application areas. Consider the following constraint on a distribution $\mathbf{R}$

$$\Pr(-x^T\mathbf{R} \le \alpha) \ge 1 - \epsilon, \tag{2}$$

for given $x$, $\alpha$ and $\epsilon \in (0, 1)$. In this case, the infimum over $\alpha$ such that (2) is satisfied is referred to as the *$\epsilon$-VaR* of $\mathbf{R}$. Within certain restricted distribution families, such as $Q$-radially symmetric distributions, (2) can be (equivalently) transformed to a deterministic second order cone constraint (depending on the range of $\epsilon$) [3]. Unfortunately, determining whether (2) can be satisfied for given $x$, $\alpha$ and $\epsilon \in (0, 1)$ is NP-hard in general [8]. Suppose however that one knew the distribution of $\mathbf{R}$ belonged to a certain family, such as $(\mu, \Sigma)$.[5] Given such knowledge, it is natural to consider whether (2) can be satisfied in a worst case sense. That is, consider

$$\left[ \inf_{\mathbf{R} \sim (\mu, \Sigma)} \Pr(-x^T \mathbf{R} \le \alpha) \right] \ge 1 - \epsilon. \tag{3}$$

Here the infimum of $\alpha$ values satisfying (3) is referred to as the *worst-case $\epsilon$-VaR*. If we have additional information about the underlying distribution, such as symmetry or unimodality, the worst-case $\epsilon$-VaR can be reduced. Importantly, using the results of the previous section, we can easily determine the worst-case $\epsilon$-VaR for various distribution families. These can also be used to provide a *tractable* bound on the $\epsilon$-VaR even when the distribution is known.

**Proposition 1** *For alternative distribution families, the worst-case $\epsilon$-VaR constraint (3) is given by:*

$$\textit{if } \mathbf{R} \sim (\mu, \Sigma) \quad \textit{then} \quad \alpha \ge -\mu_x + \sqrt{\frac{1 - \epsilon}{\epsilon}} \sigma_x, \tag{4}$$

$$\textit{if } \mathbf{R} \sim (\mu, \Sigma)_S \quad \textit{then} \quad \begin{cases} \alpha \ge -\mu_x + \sqrt{\frac{1}{2\epsilon}} \sigma_x, & \textit{if } \epsilon \in (0, \frac{1}{2}) \\ \alpha \ge -\mu_x, & \textit{if } \epsilon \in [\frac{1}{2}, 1) \end{cases} \tag{5}$$

$$\textit{if } \mathbf{R} \sim (\mu, \Sigma)_{SU} \quad \textit{then} \quad \begin{cases} \alpha \ge -\mu_x + \frac{2}{3}\sqrt{\frac{1}{2\epsilon}} \sigma_x, & \textit{if } \epsilon \in (0, \frac{1}{2}) \\ \alpha \ge -\mu_x, & \textit{if } \epsilon \in [\frac{1}{2}, 1) \end{cases} \tag{6}$$

$$\textit{if } \mathbf{R} \sim \mathcal{N}(\mu, \Sigma) \quad \textit{then} \quad \alpha \ge -\mu_x + \Phi^{-1}(1 - \epsilon)\sigma_x, \tag{7}$$

*where $\mu_x = x^T \mu$, $\sigma_x = \sqrt{x^T \Sigma x}$ and $\Phi(\cdot)$ is the c.d.f. of the standard normal distribution $\mathcal{N}(0, 1)$.*

It turns out some results of Proposition 1 are known. In fact, the first bound (4) has been extensively studied. However, given the results of the previous section, we can now provide a much simpler proof.[6] (This simplicity will also allow us to achieve some useful new bounds in Section 4 below.)

**Proof:** From Corollary 1 it follows that

$$\inf_{\mathbf{R} \sim (\mu, \Sigma)} \Pr(-x^T \mathbf{R} \le \alpha) = \inf_{\mathbf{r} \sim (-\mu_x, \sigma_x^2)} \Pr(\mathbf{r} \le \alpha) = 1 - \sup_{\mathbf{r} \sim (-\mu_x, \sigma_x^2)} \Pr(\mathbf{r} > \alpha). \tag{8}$$

Given that the problem is reduced to the univariate case, we simply exploit classical inequalities:

$$\textit{if } \mathbf{x} \sim (\mu, \sigma^2) \quad \textit{then} \quad \Pr(\mathbf{x} > t) \le \frac{\sigma^2}{\sigma^2 + (\mu - t)^2}, \tag{9}$$

$$\textit{if } \mathbf{x} \sim (\mu, \sigma^2)_S \quad \textit{then} \quad \Pr(\mathbf{x} > t) \le \frac{1}{2}\min(1, \frac{\sigma^2}{(\mu - t)^2}), \tag{10}$$

$$\textit{if } \mathbf{x} \sim (\mu, \sigma^2)_{SU} \quad \textit{then} \quad \Pr(\mathbf{x} > t) \le \frac{1}{2}\min(1, \frac{4}{9}\frac{\sigma^2}{(\mu - t)^2}), \tag{11}$$

for $t \ge \mu$.[7] Now to prove (4), simply plug (8) into (3) and notice that an application of (9) leads to

$$\alpha \ge -\mu_x \quad \text{and} \quad 1 - \frac{\sigma_x^2}{\sigma_x^2 + (-\mu_x - \alpha)^2} \ge 1 - \epsilon.$$

(4) then follows by simple rearrangement. The same procedure can be used to prove (5), (6), (7). ∎

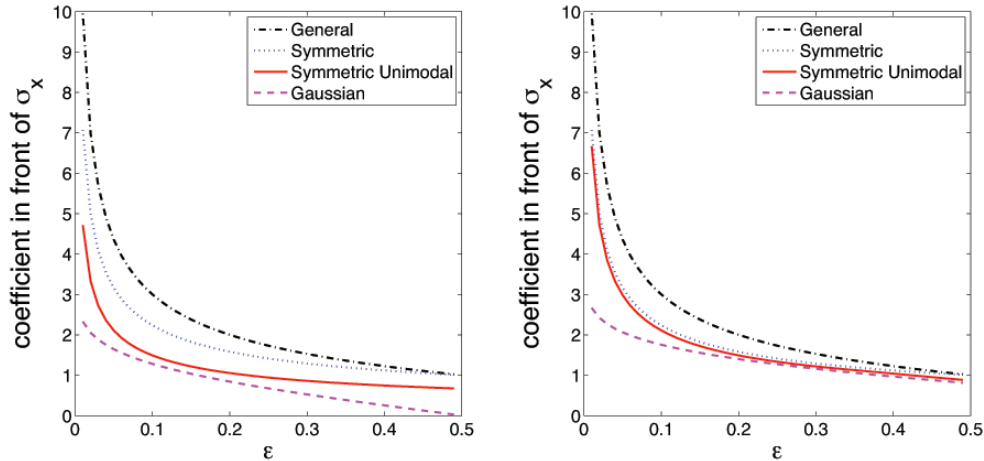

Figure 1: Comparison of the coefficients in front of $\sigma_x$ for different distribution families in Proposition 1 (left) and Proposition 2 (right). Only the range $\epsilon \in (0, \frac{1}{2})$ is depicted.

Proposition 1 clearly illustrates the benefit of prior knowledge. Figure 1 compares the coefficients on $\sigma_x$ among the different worst case VaR for different distribution families. The large gap between coefficients for general and symmetric (linear) unimodal distributions demonstrates how additional constraints can generate much less conservative solutions while still ensuring robustness.

Beyond simplifying existing proofs, Proposition 1 can be used to extend some of the uses of the VaR criterion in different application areas.

**Minimax probability classification [2]:** Lanckriet et al. [2] first studied the value-at-risk constraint in binary classification. In this scenario, one is given labeled data from two different sources and seeks a robust separating hyperplane. From the data, the distribution families $(\mu_1, \Sigma_1)$ and $(\mu_2, \Sigma_2)$ can be estimated. Then a robust hyperplane can be recovered by minimizing the worst-case error

$$\min_{x \neq 0, \alpha, \epsilon} \epsilon \text{ s.t. } \left[ \inf_{\mathbf{R}_1 \sim (\mu_1, \Sigma_1)} \Pr(x^T \mathbf{R}_1 \leq \alpha) \right] \geq 1 - \epsilon \text{ and } \left[ \inf_{\mathbf{R}_2 \sim (\mu_2, \Sigma_2)} \Pr(x^T \mathbf{R}_2 \geq \alpha) \right] \geq 1 - \epsilon, \quad (12)$$

where $x$ is the normal vector of the hyperplane, $\alpha$ is the offset and $\epsilon$ controls the error probability. Note that the results in [2] follow from using the bound (4). However, interesting additional facts arise when considering alternative distribution families. For example, consider symmetric distributions. In this case, suppose we knew in advance that the optimal $\epsilon$ lay in $[\frac{1}{2}, 1)$, meaning that no hyperplane predicts better than random guessing. Then the constraints in (12) become linear, covariance information becomes useless in determining the optimal hyperplane, and the optimization concentrates solely on separating the means of two classes. Although such a result might seem surprising, it is a direct consequence of symmetry: the worst-case distributions are forced to put probability mass arbitrarily far away on both sides of the mean, thereby eliminating any information brought by covariance. When the optimal $\epsilon$ lies in $(0, \frac{1}{2})$, however, covariance information becomes meaningful, since the worst-case distributions can no longer put probability mass arbitrarily far away on both sides of the mean (owing to the existence of a hyperplane that predicts labels better than random guessing). In this case, the optimization problems involving $(\mu, \Sigma)_S$ and $(\mu, \Sigma)_{SU}$ are equivalent to that for $(\mu, \Sigma)$ except that the maximum error probability $\epsilon$ becomes smaller, which is to be expected since more information about the marginal distributions should make one more confident to predict the labels of future data.

**Chance Constrained Linear Programming (CCLP) [3]:** Consider a linear program $\min_x a^T x$ s.t. $r^T x \geq 0$. If the coefficient $r$ is uncertain, it is clear that solving the linear program merely using the expected value of $r$ could result in a solution $x$ that was sub-optimal or even infeasible. Calafiore and El Ghaoui studied this problem in [3], and imposed the inequality constraint with high probability, leading to the the so-called chance constrained linear program (CCLP):

$$\min_x a^T x \text{ s.t. } \left[ \inf_{\mathbf{R} \sim (\mu, \Sigma)} \Pr(-x^T \mathbf{R} \leq 0) \right] \geq 1 - \epsilon. \quad (13)$$

In this case, $\alpha$ is simply 0 and $\epsilon$ is given by the user. Depending on the value of $\epsilon$, the chance constraint can be equivalently transformed into a second order cone constraint or a linear constraint. The work in [3] concentrates on the general and symmetric distribution families. In the latter case, [3] uses the first part of inequality (5) as a sufficient condition for guaranteeing robust solutions. Note however that from Corollary 1 and Proposition 1 one can now see that (5) is also a necessary condition. Although the symmetric linear unimodal case is not discussed in [3], from Proposition 1 again one can see that incorporating bound (6) in (13) yields a looser constraint than does (5), hence the feasible region will be enlarged and the optimum value of the CCLP potentially reduced, corresponding to the intuition that increased prior knowledge leads to more optimized results.

**Portfolio Selection [4]:** In portfolio selection, let $\mathbf{R}$ represent the (uncertain) returns of a suite of financial assets, and $x$ the weighting one would like to put on the various assets. Here $\alpha > -x^T\mathbf{R}$ represents an upper bound on the loss one might suffer with weighting $x$. The goal is to minimize an upper bound on the loss that holds with high probability,[8] say $1-\epsilon$, specified by the user

$$\min_{x,\alpha} \alpha \text{ s.t. } \left[\inf_{\mathbf{R}\sim(\mu,\Sigma)} \Pr(-x^T\mathbf{R} \le \alpha)\right] \ge 1-\epsilon. \tag{14}$$

This criterion has been studied by El Ghaoui et al. [4] in the worst case setting. Previous work has not addressed the case when additional symmetry or linear unimodal information is available. However, comparing the minimal value of $\alpha$ in Proposition 1, we see that such additional information, such as symmetry or unimodality, indeed decreases our potential loss, as shown clearly in Figure 1. This makes sense, since the more one knows about uncertain returns the less risk one should have to bear. Note also that when incorporating additional information, the optimal portfolio, represented by $x$, is changed as well but remains mean-variance efficient when $\epsilon \in (0, \frac{1}{2})$.

**Uncertain MDPs with reward uncertainty:** The standard planning problem in Markov decision processes (MDPs) is to find a policy such that maximizes the expected total discounted return. This nonlinear optimization problem can be efficiently solved by dynamic programming, provided that the model parameters (transition kernel and reward function) are exactly known. Unfortunately, this is rarely the case in practice. Delage and Mannor [8] extend this problem to the uncertain case by employing the value-at-risk type constraint (2) and assuming the unknown reward model and transition kernel are drawn from a known distribution (Gaussian and Dirichlet respectively). Unfortunately, [8] also proves that the constraint (2) is generally NP-hard to satisfy unless one assumes some very restricted form of distribution, such as Gaussian. Alternatively, note that one can use the *worst case* value-at-risk formulation (3) to obtain a tractable approximation to (2)

$$\min_{x,\alpha} \alpha \text{ s.t. } \left[\inf_{\mathbf{R}\sim(\mu,\Sigma)} \Pr(-x^T\mathbf{R} \le \alpha)\right] \ge 1-\epsilon, \tag{15}$$

where $\mathbf{R}$ is the reward function (unknown but assumed to belong to $(\mu, \Sigma)$) and $x$ represents a discounted-stationary state-action visitation distribution (which can be used to recover an optimal behavior policy). Although this worst case formulation (15) might appear to be conservative compared to working with a known distribution on $\mathbf{R}$ and using (2), when additional information about the distribution is available, such as symmetry or unimodality, (15) can be brought very close to using a Gaussian distribution, as shown in Figure 1. Thus, given reasonable constraints, the minimax approach does not have to be overly conservative, while providing robustness and tractability.

## 4  Application to Worst-case *Conditional* Value-at-risk

Finally, we investigate the more refined conditional value-at-risk (CVaR) criterion that bounds the *conditional expectation* of losses beyond the value-at-risk (VaR). This criterion has been of growing prominence in many areas recently. Consider the following quantity defined as the mean of a tail distribution:

$$\hat{f} = \mathbb{E}\left[-x^T\mathbf{R} \mid \Pr(-x^T\mathbf{R} \le \alpha^*) \ge 1-\epsilon\right] \text{ where } \alpha^* = \arg\min_\alpha \alpha \text{ s.t. } \Pr(-x^T\mathbf{R} \le \alpha) \ge 1-\epsilon. \tag{16}$$

Here, $\alpha^*$ is the *value-at-risk* and $\hat{f}$ is the *conditional value-at-risk* of $\mathbf{R}$. It is well-known that the CVaR, $\hat{f}$, is always an upper bound on the VaR, $\alpha^*$. Although it might appear that dealing with

the CVaR criterion entails greater complexity than the VaR, since VaR is directly involved in the definition of CVaR, it turns out that CVaR can be more directly expressed as

$$\hat{f} = \min_{\alpha} \quad \alpha + \frac{1}{\epsilon}\mathbb{E}\big[(-x^T\mathbf{R} - \alpha)_+\big], \qquad (17)$$

where $(x)_+ = \max(0, x)$ [14]. Unlike the VaR constraint (2), (17) is always (jointly) convex in $x$ and $\alpha$. Thus if $\mathbf{R}$ were discrete, $\hat{f}$ could be easily computed by a linear program [14; 5]. However, the expectation in (17) involves a high dimensional integral in general, whose analytical solution is not always available, thus $\hat{f}$ is still hard to compute in practice. Although one potential remedy might be to use Monte Carlo techniques to approximate the expectation, we instead take a robust approach: As before, suppose one knew the distribution of $\mathbf{R}$ belonged to a certain family, such as $(\mu, \Sigma)$. Given such knowledge, it is natural to consider the *worst-case* CVaR

$$f = \sup_{\mathbf{R}\sim(\mu,\Sigma)} \min_{\alpha} \quad \alpha + \frac{1}{\epsilon}\mathbb{E}\big[(-x^T\mathbf{R} - \alpha)_+\big] = \min_{\alpha} \quad \sup_{\mathbf{R}\sim(\mu,\Sigma)} \alpha + \frac{1}{\epsilon}\mathbb{E}\big[(-x^T\mathbf{R} - \alpha)_+\big], \quad (18)$$

where the interchangeability of the min and sup operators follows from the classic minimax theorem [15]. Importantly, as in the previous section, we can determine the worst-case CVaR for various distribution families. If one has additional information about the underlying distribution, such as symmetry or unimodality, the worst-case CVaR can be reduced. These can be used to provide a tractable bound on the CVaR even when the distribution is known.

**Proposition 2** *For alternative distribution families, the worst-case CVaR is given by:*

$$\text{if } \mathbf{R} \sim (\mu, \Sigma) \text{ then } \alpha = -\mu_x + \frac{(2\epsilon - 1)}{2\sqrt{\epsilon(1-\epsilon)}}\sigma_x, \quad f = -\mu_x + \sqrt{\frac{1-\epsilon}{\epsilon}}\sigma_x, \qquad (19)$$

$$\text{if } \mathbf{R} \sim (\mu, \Sigma)_S \text{ then } \begin{cases} \alpha = -\mu_x + \frac{1}{\sqrt{8\epsilon}}\sigma_x, \ f = -\mu_x + \frac{1}{\sqrt{2\epsilon}}\sigma_x & \text{if } \epsilon \in (0, \frac{1}{2}] \\ \alpha = -\mu_x - \frac{1}{\sqrt{8(1-\epsilon)}}\sigma_x, \ f = -\mu_x + \frac{\sqrt{1-\epsilon}}{\sqrt{2\epsilon}}\sigma_x & \text{if } \epsilon \in [\frac{1}{2}, 1) \end{cases} \qquad (20)$$

$$\text{if } \mathbf{R} \sim (\mu, \Sigma)_{SU} \text{ then } \begin{cases} \alpha = -\mu_x + \frac{1}{3\sqrt{\epsilon}}\sigma_x, \ f = -\mu_x + \frac{2}{3\sqrt{\epsilon}}\sigma_x & \text{if } \epsilon \in (0, \frac{1}{3}] \\ \alpha = -\mu_x + \sqrt{3}(1-2\epsilon)\sigma_x, \ f = -\mu_x + \sqrt{3}(1-\epsilon)\sigma_x & \text{if } \epsilon \in [\frac{1}{3}, \frac{2}{3}] \\ \alpha = -\mu_x - \frac{1}{3\sqrt{1-\epsilon}}\sigma_x, \ f = -\mu_x + \frac{2\sqrt{1-\epsilon}}{3\epsilon}\sigma_x, & \text{if } \epsilon \in [\frac{2}{3}, 1) \end{cases} \qquad (21)$$

$$\text{if } \mathbf{R} \sim \mathcal{N}(\mu, \Sigma) \text{ then } f = -\mu_x + \frac{e^{-\frac{(\Phi^{-1}(1-\epsilon))^2}{2}}}{\sqrt{2\pi}\epsilon}\sigma_x, \qquad (22)$$

*where $\mu_x = x^T\mu$, $\sigma_x = \sqrt{x^T\Sigma x}$ and $\Phi(\cdot)$ is the c.d.f. of a standard normal distribution $\mathcal{N}(0,1)$.*

The results of Proposition 2 are a novel contribution of this paper, with the exception of (22), which is a standard result in stochastic programming [7].

**Proof:** We know from Corollary 1 that

$$\sup_{\mathbf{R}\sim(\mu,\Sigma)} \mathbb{E}\big[(-x^T\mathbf{R} - \alpha)_+\big] = \sup_{\mathbf{r}\sim(-\mu_x, \sigma_x^2)} \mathbb{E}\big[(\mathbf{r} - \alpha)_+\big], \qquad (23)$$

which reduces the problem to the univariate case. To proceed, we will need to make use of the univariate results given in Proposition 3 below. Assuming Proposition 3 for now, we show how to prove (19): In this case, substitute (23) into (18) and apply (24) from Proposition 3 below to obtain

$$f = \min_{\alpha} \quad \alpha + \frac{1}{2\epsilon}\Big[(-\mu_x - \alpha) + \sqrt{\sigma_x^2 + (-\mu_x - \alpha)^2}\Big].$$

This is a convex univariate optimization problem in $\alpha$. Taking the derivative with respect to $\alpha$ and setting to zero gives $\alpha = -\mu_x + \frac{(2\epsilon-1)}{2\sqrt{\epsilon(1-\epsilon)}}\sigma_x$. Substituting back we obtain $f = -\mu_x + \sqrt{\frac{1-\epsilon}{\epsilon}}\sigma_x$. A similar strategy can be used to prove (20), (21), and (22). ∎

As with Proposition 1, Proposition 2 illustrates the benefit of prior knowledge. Figure 1 (right) compares the coefficients on $\sigma_x$ among different worst-case CVaR quantities for different families. Comparing VaR and CVaR in Figure 1 shows that unimodality has less impact on improving CVaR.

A key component of Proposition 2 is its reliance on the following important univariate results. The following proposition gives tight bounds of the expected survival function for the various families.

**Proposition 3** *For alternative distribution families, the expected univariate survival functions are:*

$$\sup_{\mathbf{x}\sim(\mu,\sigma^2)} \mathbb{E}\big[(\mathbf{x}-t)_+\big] = \frac{1}{2}\Big[(\mu-t)+\sqrt{\sigma^2+(\mu-t)^2}\Big], \tag{24}$$

$$\sup_{\mathbf{x}\sim(\mu,\sigma^2)_S} \mathbb{E}\big[(\mathbf{x}-t)_+\big] = \begin{cases} \frac{\sigma-t+\mu}{2}, & \text{if } \mu-\frac{\sigma}{2} \le t \le \mu+\frac{\sigma}{2} \\ \frac{\sigma^2}{8(t-\mu)}, & \text{if } t > \mu+\frac{\sigma}{2} \\ -\frac{\sigma^2+8(t-\mu)^2}{8(t-\mu)}, & \text{if } t < \mu-\frac{\sigma}{2} \end{cases} \tag{25}$$

$$\sup_{\mathbf{x}\sim(\mu,\sigma^2)_{SU}} \mathbb{E}\big[(\mathbf{x}-t)_+\big] = \begin{cases} \frac{(\sqrt{3}\sigma-t+\mu)^2}{4\sqrt{3}\sigma}, & \text{if } \mu-\frac{\sigma}{\sqrt{3}} \le t \le \mu+\frac{\sigma}{\sqrt{3}} \\ \frac{\sigma^2}{9(t-\mu)}, & \text{if } t > \mu+\frac{\sigma}{\sqrt{3}} \\ -\frac{\sigma^2+9(t-\mu)^2}{9(t-\mu)}, & \text{if } t < \mu-\frac{\sigma}{\sqrt{3}} \end{cases} \tag{26}$$

Here (26) is a further novel contribution of this paper. Proofs of (24) and (25) can be found in [1].

Interestingly, to the best of our knowledge, the worst-case CVaR criterion has not yet been applied to any of the four problems mentioned in the previous section[9]. Given the space constraints, we can only discuss the direct application of worst-case CVaR to the portfolio selection problem. We note that CVaR has been recently applied to $\nu$-SVM learning in [16].

**Implications for Portfolio Selection:** By comparing Propositions 1 and 2, the first interesting conclusion one can reach about portfolio selection is that, without considering any additional information, the worst-case CVaR criterion yields the same optimal portfolio weighting $x$ as the worst-case VaR criterion (recall that VaR minimizes $\alpha$ in Proposition 1 by adjusting $x$ while CVaR minimizes $f$ by adjusting $x$ in Proposition 2). However, the worst-case distributions for the two approaches are not the same, which can be seen from the relation (16) between VaR and CVaR and observing that $\alpha$ in (4) is not the same as in (19). Next, when additional symmetry information is taken into account and $\epsilon \in (0, \frac{1}{2})$, CVaR and VaR again select the same portfolio but under different worst-case distributions. When unimodality is added, the CVaR criterion finally begins to select different portfolios than VaR.

## 5 Concluding Remarks

We have provided a simpler yet broader proof of the general linear projection property for distribution families with given mean and covariance. The proof strategy can be easily extended to more restricted distribution families. A direct implication of our results is that worst-case analyses of multivariate expectations can often be reduced to those of univariate ones. By combining this trick with classic univariate inequalities, we were able to provide worst-case analyses of two widely adopted constraints (based on value-at-risk criteria). Our analysis recovers some existing results in a simpler way while also provides new insights on incorporating additional information.

Above, we assumed the first and second moments of the underlying distribution were precisely known, which of course is questionable in practice. Fortunately, there are standard techniques for handling such additional uncertainty. One strategy, proposed in [2], is to construct a (bounded and convex) uncertainty set $\mathcal{U}$ over $(\mu, \Sigma)$, and then applying a similar minimax formulation but with respect to $(\mu, \Sigma) \in \mathcal{U}$. As shown in [2], appropriately chosen uncertainty sets amount to adding straightforward regularizations to the original problem. A second approach is simply to lower one's confidence of the constraints and rely on the fact that the moment estimates are close to their true values within some additional confidence bound [17]. That is, instead of enforcing the constraint (3) or (18) *surely*, one can instead plug-in the estimated moments and argue that constraints will be satisfied within some diminished probability. For an application of this strategy in CCLP, see [3].

## Acknowledgement

We gratefully acknowledge support from the Alberta Ingenuity Centre for Machine Learning, the Alberta Ingenuity Fund, iCORE and NSERC. Csaba Szepesvàri is on leave from MTA SZTAKI, Bp. Hungary.

## Footnotes

[1] In preparing the final version of this paper, we noticed that a very recent work [9] proved the one dimensional case by a similar technique as ours.

[2]A sufficient but not necessary condition for log-concavity is having log-concave densities. This can be used to verify log-concavity of normal and uniform distributions. In the univariate case, log-concave distributions are called strongly unimodal, which is only a *proper* subset of univariate unimodal distributions [10].

[3]If $X$ is a vector we can also extend this theorem to other multivariate unimodalities such as symmetric star/block/convex unimodal.

[4]The closure of $(\mu, \Sigma)$, $(\mu, \Sigma)_S$, $(\mu, \Sigma)_L$, and $(\mu, \Sigma)_{SU}$ under linear projection is critical for Corollary 1 to hold. Corollary 1 fails for other kinds of multivariate unimodalities, such as symmetric star/block/convex unimodal. It also fails for $(\mu, \Sigma)_+$, a distribution family whose support is contained in the nonnegative orthant. This is not surprising since determining whether the set $(\mu, \Sigma)_+$ is empty is already NP-hard [11].

[5]We will return to the question of when such moment information is also subject to uncertainty in Section 5.

[6][2] and [3] provide a proof of (4) based on the multivariate Chebyshev inequality in [12]; [4] proves (4) from dual optimality; and the proof in [6] utilizes two point support property of the general constraint (3).

[7](9) is known as the (one-sided) Chebyshev inequality. Two-sided version of (11) is known as the Gauss inequality. These classical bounds are tight. Proofs can be found in [13], for example.

[8]Note that seeking to minimize the loss *surely* leads to a meaningless outcome. For example, if $\epsilon = 0$, the optimization problem trivially says that the loss of any portfolio will be no larger than $\infty$.

[9]Except the very recent work of [9] on portfolio selection.

# References

[1] R. Jagannathan. "Minimax procedure for a class of linear programs under uncertainty". *Operations Research*, vol. 25(1):pp. 173–177, 1977.

[2] Gert R.G. Lanckriet, Laurent El Ghaoui, Chiranjib Bhattacharyya and Michael I. Jordan. "A robust minimax approach to classification". *Journal of Machine Learning Research*, vol. 03:pp. 555–582, 2002.

[3] G.C.Calafiore and Laurent El Ghaoui. "On distributionally robust chance-constrained linear programs". *Journal of Optimization Theory and Applications*, vol. 130(1):pp. 1–22, 2006.

[4] Laurent El Ghaoui, Maksim Oks and Francois Oustry. "Worst-case value-at-risk and robust portfolio optimization: a conic programming approach". *Operations Research*, vol. 51(4):pp. 542–556, 2003.

[5] Shu-Shang Zhu and Masao Fukushima. "Worst-case conditional value-at-risk with application to robust portfolio management". *Operations Research*, vol. 57(5):pp. 1155–1168, 2009.

[6] Ioana Popescu. "Robust mean-covariance solutions for stochastic optimization". *Operations Research*, vol. 55(1):pp. 98–112, 2007.

[7] András Prékopa. *Stochastic Programming*. Springer, 1995.

[8] Erick Delage and Shie Mannor. "Percentile optimization for Markov decision processes with parameter uncertainty". *Operations Research*, to appear 2009.

[9] Li Chen, Simai He and Shuzhong Zhang. "Tight Bounds for Some Risk Measures, with Applications to Robust Portfolio Selection". Tech. rep., Department of Systerms Engineering and Engineering Management, The Chinese University of Hongkong, 2009.

[10] Sudhakar Dharmadhikari and Kumar Joag-Dev. *Unimodality, Convexity, and Applications*. Academic Press, 1988.

[11] Dimitris Bertsimas and Ioana Popescu. "Optimal inequalities in probability theory a convex optimization approach". *SIAM Journal on Optimization*, vol. 15(3):pp. 780–804, 2005.

[12] Albert W. Marshall and Ingram Olkin. "Multivariate Chebyshev inequalities". *Annals of Mathematical Statistics*, vol. 31(4):pp. 1001–1014, 1960.

[13] Ioana Popescu. "A semidefinite programming approach to optimal moment bounds for convex classes of distributions". *Mathematics of Operations Research*, vol. 30(3):pp. 632–657, 2005.

[14] R. Tyrrell Rockafellar and Stanislav Uryasev. "Optimization of conditional value-at-risk". *Journal of Risk*, vol. 2(3):pp. 493–517, 2000.

[15] Ky Fan. "Minimax Theorems". *Proceedings of the National Academy of Sciences*, vol. 39(1):pp. 42–47, 1953.

[16] Akiko Takeda and Masashi Sugiyama. "$\nu$-support vector machine as conditional value-at-risk minimization". In *Proceedings of the $25^{th}$ International Conference on Machine Learning*, pp. 1056–1063. 2008.

[17] John Shawe-Taylor and Nello Cristianini. "Estimating the moments of a random vector with applications". In *Proceedings of GRETSI 2003 Conference*, pp. 47–52. 2003.

